# Testing a Bayesian Measure of Representativeness Using a Large Image Database

**Joshua T. Abbott**
Department of Psychology
University of California, Berkeley
Berkeley, CA 94720
joshua.abbott@berkeley.edu

**Katherine A. Heller**
Department of Brain and Cognitive Sciences
Massachusetts Institute of Technology
Cambridge, MA 02139
kheller@mit.edu

**Zoubin Ghahramani**
Department of Engineering
University of Cambridge
Cambridge, CB2 1PZ, U.K.
zoubin@eng.cam.ac.uk

**Thomas L. Griffiths**
Department of Psychology
University of California, Berkeley
Berkeley, CA 94720
tom_griffiths@berkeley.edu

## Abstract

How do people determine which elements of a set are most representative of that set? We extend an existing Bayesian measure of representativeness, which indicates the representativeness of a sample from a distribution, to define a measure of the representativeness of an item to a set. We show that this measure is formally related to a machine learning method known as Bayesian Sets. Building on this connection, we derive an analytic expression for the representativeness of objects described by a sparse vector of binary features. We then apply this measure to a large database of images, using it to determine which images are the most representative members of different sets. Comparing the resulting predictions to human judgments of representativeness provides a test of this measure with naturalistic stimuli, and illustrates how databases that are more commonly used in computer vision and machine learning can be used to evaluate psychological theories.

## 1   Introduction

The notion of "representativeness" appeared in cognitive psychology as a proposal for a heuristic that people might use in the place of performing a probabilistic computation [1, 2]. For example, we might explain why people believe that the sequence of heads and tails HHTHT is more likely than HHHHH to be produced by a fair coin by saying that the former is more representative of the output of a fair coin than the latter. This proposal seems intuitive, but raises a new problem: How is representativeness itself defined? Various proposals have been made, connecting representativeness to existing quantities such as similarity [1] (itself an ill-defined concept [3]), or likelihood [2]. Tenenbaum and Griffiths [4] took a different approach to this question, providing a "rational analysis" of representativeness by trying to identify the problem that such a quantity solves. They proposed that one sense of representativeness is being a good example of a concept, and then showed how this could be quantified via Bayesian inference. The resulting model outperformed similarity and likelihood in predicting human representativeness judgments for two kinds of simple stimuli.

In this paper, we extend this definition of representativeness, and provide a more comprehensive test of this account using naturalistic stimuli. The question of what makes a good example of a concept is of direct relevance to computer scientists as well as cognitive scientists, providing a way to build better systems for retrieving images or documents relevant to a user's query. However, the

model presented by Tenenbaum and Griffiths [4] is overly restrictive in requiring the concept to be pre-defined, and has not been tested in the context of a large-scale information retrieval system. We extend the Bayesian measure of representativeness to apply to the problem of deciding which objects are good examples of a set of objects, show that the resulting model is closely mathematically related to an existing machine learning method known as Bayesian Sets [5], and compare this model to similarity and likelihood as an account of people's judgments of the extent to which images drawn from a large database are representative of different concepts. In addition, we show how measuring the representativeness of items in sets can also provide a novel method of finding outliers in sets.

By extending the Bayesian measure of representativeness to apply to sets of objects and testing it with a large image database, we are taking the first steps towards a closer integration of the methods of cognitive science and machine learning. Cognitive science experiments typically use a small set of artificial stimuli, and evaluate different models by comparing them to human judgments about those stimuli. Machine learning makes use of large datasets, but relies on secondary sources of "cognitive" input, such as the labels people have applied to images. We combine these methods by soliciting human judgments to test cognitive models with a large set of naturalistic stimuli. This provides the first experimental comparison of the Bayesian Sets algorithm to human judgments, and the first evaluation of the Bayesian measure of representativeness in a realistic applied setting.

The plan of the paper is as follows. Section 2 provides relevant background information, including psychological theories of representativeness and the definition of Bayesian Sets. Section 3 then introduces our extended measure of representativeness, and shows how it relates to Bayesian Sets. Section 4 describes the dataset derived from a large image database that we use for evaluating this measure, together with the other psychological models we use for comparison. Section 5 presents the results of an experiment soliciting human judgments about the representativeness of different images. Section 6 provides a second form of evaluation, focusing on identifying outliers from sets. Finally, Section 7 concludes the paper.

## 2 Background

To approach our main question of which elements of a set are most representative of that set, we first review previous psychological models of representativeness with a particular focus on the rational model proposed by Tenenbaum and Griffiths [4]. We then introduce Bayesian Sets [5].

### 2.1 Representativeness

While the notion of representativeness has been most prominent in the literature on judgment and decision-making, having been introduced by Kahneman and Tversky [1], similar ideas have been explored in accounts of human categorization and inductive inference [6, 7]. In these accounts, representativeness is typically viewed as a form of similarity between an outcome and a process or an object and a concept. Assume some data $d$ has been observed, and we want to evaluate its representativeness of a hypothesized process or concept $h$. Then $d$ is representative of $h$ if it is similar to the observations $h$ typically generates. Computing similarity requires defining a similarity metric. In the case where we want to evaluate the representativeness of an outcome to a set, we might use metrics of the kind that are common in categorization models: an exemplar model defines similarity in terms of the sum of the similarities to the other objects in the set (e.g., [8, 9]), while a prototype model defines similarity in terms of the similarity to a prototype that captures the characteristics of the set (e.g., [10]).

An alternative to similarity is the idea that representativeness might track the likelihood function $P(d|h)$ [11]. The main argument for this proposed equivalence is that the more frequently $h$ leads to observing $d$, the more representative $d$ should be of $h$. However, people's judgments from the coin flip example with which we started the paper go against this idea of equivalence, since both flips have equal likelihood yet people tend to judge HHTHT as more representative of a fair coin. Analyses of typicality have also argued against the adequacy of frequency for capturing people's judgments about what makes a good example of a category [6].

Tenenbaum and Griffiths [4] took a different approach to this question, asking what problem representativeness might be solving, and then deriving an optimal solution to that problem. This approach is similar to that taken in Shepard's [12] analysis of generalization, and to Anderson's [13] idea of

rational analysis. The resulting rational model of representativeness takes the problem to be one of selecting a good example, where the best example is the one that best provides evidence for the target process or concept relative to possible alternatives. Given some observed data $d$ and a set of of hypothetical sources, $\mathcal{H}$, we assume that a learner uses Bayesian inference to infer which $h \in \mathcal{H}$ generated $d$. Tenenbaum and Griffiths [4] defined the representativeness of $d$ for $h$ to be the evidence that $d$ provides in favor of a specific $h$ relative to its alternatives,

$$R(d,h) = \log \frac{P(d|h)}{\sum_{h' \neq h} P(d|h')P(h')}, \tag{1}$$

where $P(h')$ in the denominator is the prior distribution on hypotheses, re-normalized over $h' \neq h$.

## 2.2 Bayesian Sets

If given a small set of items such as "ketchup", "mustard", and "mayonnaise" and asked to produce other examples that fit into this set, one might give examples such as "barbecue sauce", or "honey". This task is an example of clustering on-demand, in which the original set of items represents some concept or cluster such as "condiment" and we are to find other items that would fit appropriately into this set. Bayesian Sets is a formalization of this process in which items are ranked by a model-based probabilistic scoring criterion, measuring how well they fit into the original cluster [5].

More formally, given a data collection $\mathcal{D}$, and a subset of items $\mathcal{D}_s = \{\mathbf{x}_1, \ldots, \mathbf{x}_N\} \subset \mathcal{D}$ representing a concept, the Bayesian Sets algorithm ranks an item $\mathbf{x}^* \in \{\mathcal{D} \setminus \mathcal{D}_s\}$ by the following scoring criterion

$$\text{Bscore}(\mathbf{x}^*) = \frac{p(\mathbf{x}^*, \mathcal{D}_s)}{p(\mathbf{x}^*)p(\mathcal{D}_s)} \tag{2}$$

This ratio intuitively compares the probability that $\mathbf{x}^*$ and $\mathcal{D}_s$ were generated by some statistical model with the same, though unknown, model parameters $\theta$, versus the probability that $\mathbf{x}^*$ and $\mathcal{D}_s$ were generated by some statistical model with different model parameters $\theta_1$ and $\theta_2$.

Each of the three terms in Equation 2 are marginal likelihoods and can be expressed as the following integrals over $\theta$ since the model parameter is assumed to be unknown: $p(\mathbf{x}^*) = \int p(\mathbf{x}^*|\theta)p(\theta)d\theta$, $p(\mathcal{D}_s) = \int \left[\prod_{n=1}^{N} p(\mathbf{x}_n|\theta)\right] p(\theta)d\theta$, and $p(\mathbf{x}^*, \mathcal{D}_s) = \int \left[\prod_{n=1}^{N} p(\mathbf{x}_n|\theta)\right] p(\mathbf{x}^*|\theta)p(\theta)d\theta$.

For computational efficiency reasons, Bayesian Sets is typically run on binary data. Thus, each item in the data collection, $\mathbf{x}_i \in \mathcal{D}$, is represented as a binary feature vector $\mathbf{x}_i = (x_{i1}, \ldots, x_{iJ})$ where $x_{ij} \in \{0, 1\}$, and defined under a model in which each element of $\mathbf{x}_i$ has an independent Bernoulli distribution $p(\mathbf{x}_i|\theta) = \prod_j \theta_j^{x_{ij}}(1-\theta_j)^{1-x_{ij}}$ and conjugate Beta prior $p(\theta|\alpha, \beta) = \prod_j \frac{\Gamma(\alpha_j+\beta_j)}{\Gamma(\alpha_j)\Gamma(\beta_j)} \theta_j^{\alpha_j-1}(1-\theta_j)^{\beta_j-1}$. Under these assumptions, the scoring criterion for Bayesian Sets reduces to

$$\text{Bscore}(\mathbf{x}^*) = \frac{p(\mathbf{x}^*, \mathcal{D}_s)}{p(\mathbf{x}^*)p(\mathcal{D}_s)} = \prod_j \frac{\alpha_j + \beta_j}{\alpha_j + \beta_j + N} \left(\frac{\tilde{\alpha}_j}{\alpha_j}\right)^{x_{*j}} \left(\frac{\tilde{\beta}_j}{\beta_j}\right)^{1-x_{*j}} \tag{3}$$

where $\tilde{\alpha}_j = \alpha_j + \sum_{n=1}^{N} x_{nj}$ and $\tilde{\beta}_j = \beta_j + N - \sum_{n=1}^{N} x_{nj}$. The logarithm of this score is linear in $\mathbf{x}$ and can be computed efficiently as

$$\log \text{Bscore}(\mathbf{x}^*) = c + \sum_j s_j x_{*j} \tag{4}$$

where $c = \sum_j \log(\alpha_j + \beta_j) - \log(\alpha_j + \beta_j + N) + \log \tilde{\beta}_j - \log \beta_j$, $s_j = \log \tilde{\alpha}_j - \log \alpha_j - \log \tilde{\beta}_j + \log \beta_j$, and $x_{*j}$ is the $j$th component of $\mathbf{x}^*$.

The Bayesian Sets method has been tested with success on numerous datasets, over various applications including content-based image retrieval [14] and analogical reasoning with relational data [15]. Motivated by this method, we now turn to extending the previous measure of representativeness for a sample from a distribution, to define a measure of representativeness for an item to a set.

# 3 A Bayesian Measure of Representativeness for Sets of Objects

The Bayesian measure of representativeness introduced by Tenenbaum and Griffiths [4] indicated the representativeness of data $d$ for a hypothesis $h$. However, in many cases we might not know what statistical hypothesis best describes the concept that we want to illustrate through an example. For instance, in an image retrieval problem, we might just have a set of images that are all assigned to the same category, without a clear idea of the distribution that characterizes that category. In this section, we show how to extend the Bayesian measure of representativeness to indicate the representativeness of an element of a set, and how this relates to the Bayesian Sets method summarized above.

Formally, we have a set of data $\mathcal{D}_s$ and we want to know how representative an element $d$ of that set is of the whole set. We can perform an analysis similar to that given for the representativeness of $d$ to a hypothesis, and obtain the expression

$$R(d, \mathcal{D}_s) = \frac{P(d|\mathcal{D}_s)}{\sum_{\mathcal{D}' \neq \mathcal{D}_s} P(d|\mathcal{D}')P(\mathcal{D}')} \tag{5}$$

which is simply Equation 1 with hypotheses replaced by datasets. The quantities that we need to compute to apply this measure, $P(d|\mathcal{D}_s)$ and $P(\mathcal{D}')$, we obtain by marginalizing over all hypotheses. For example, $P(d|\mathcal{D}_s) = \sum_h P(d|h)P(h|\mathcal{D}_s)$, being the posterior predictive distribution associated with $\mathcal{D}_s$. If the hypotheses correspond to the continuous parameters of a generative model, then this is better expressed as $P(d|\mathcal{D}_s) = \int P(d|\theta)P(\theta|\mathcal{D}_s)$.

In the case where the set of possible datasets that is summed over in the denominator is large, this denominator will approximate $\sum_{\mathcal{D}'} P(d|\mathcal{D}')P(\mathcal{D}')$, which is just $P(d)$. This allows us to observe that this measure of representativeness will actually closely approximate the logarithm of the quantity Bscore produced by Bayesian Sets for the dataset $\mathcal{D}_s$, with

$$R(d, \mathcal{D}_s) = \log \frac{P(d|\mathcal{D}_s)}{\sum_{\mathcal{D}' \neq \mathcal{D}_s} P(d|\mathcal{D}')P(\mathcal{D}')} \approx \log \frac{P(d|\mathcal{D}_s)}{P(d)} = \log \frac{P(d, \mathcal{D}_s)}{P(d)P(\mathcal{D}_s)} = \log \mathrm{Bscore}(d)$$

This relationship provides a link between the cognitive science literature on representativeness and the machine learning literature on information retrieval, and a new way to evaluate psychological models of representativeness.

# 4 Evaluating Models of Representativeness Using Image Databases

Having developed a measure of the representativeness of an item in a set of objects, we now focus on the problem of evaluating this measure. The evaluation of psychological theories has historically tended to use simple artificial stimuli, which provide precision at the cost of ecological validity. In the case of representativeness, the stimuli previously used by Tenenbaum and Griffiths [4] to evaluate different representativeness models consisted of 4 coin flip sequences and 45 arguments based on predicates applied to a set of 10 mammals. One of the aims of this paper is to break the general trend of using such restricted kinds of stimuli, and the formal relationship between our rational model and Bayesian Sets allows us to do so. Any dataset that can be represented as a sparse binary matrix can be used to test the predictions of our measure.

We formulate our evaluation problem as one of determining how representative an image is of a labeled set of images. Using an existing image database of naturalistic scenes, we can better test the predictions of different representativeness theories with stimuli much more in common with the environment humans naturally confront. In the rest of this section, we present the dataset used for evaluation and outline the implementations of existing models of representativeness we compare our rational Bayesian model against.

## 4.1 Dataset

We use the dataset presented in [14], a subset of images taken from the Corel database commonly used in content-based image retrieval systems. The images in the dataset are partitioned into 50 labeled sets depicting unique categories, with varying numbers of images in each set (the mean is 264). The dataset is of particular interest for testing models of representativeness as each image

**Algorithm 1** Representativeness Framework

---

    **input:** a set of items, $\mathcal{D}_w$, for a particular category label $w$
    **for** each item $\mathbf{x}_i \in \mathcal{D}_w$ **do**
      let $\mathcal{D}_{wi} = \{\mathcal{D}_w \setminus \mathbf{x}_i\}$
      compute    score$(\mathbf{x}_i, \mathcal{D}_{wi})$
    **end for**
    rank items in $\mathcal{D}_w$ by this score
    **output:** ranked list of items in $\mathcal{D}_w$

---

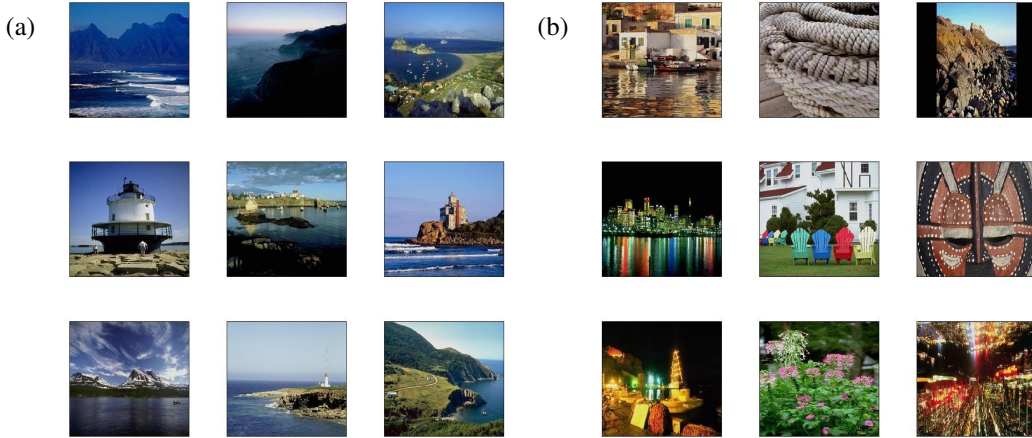

Figure 1: Results of the Bayesian model applied to the set labelled *coast*. (a) The top nine ranked images. (b) The bottom nine ranked images.

from the Corel database comes with multiple labels given by human judges. The labels have been criticized for not always being of high quality [16], which provides an additional (realistic) challenge for the models of representativeness that we aim to evaluate.

The images in this dataset are represented as 240-dimensional feature vectors, composed of 48 Gabor texture features, 27 Tamura texture features, and 165 color histogram features. The images were additionally preprocessed through a binarization stage, transforming the entire dataset into a sparse binary matrix that represents the features which most distinguish each image from the rest of the dataset. Details of the construction of this feature representation are presented in [14].

## 4.2 Models of Representativeness

We compare our Bayesian model against a likelihood model and two similarity models: a prototype model and an exemplar model. We build upon a simple leave-one-out framework to allow a fair comparison of these different representativeness models. Given a set of images with a particular category label, we iterate through each image in the set and compute a score for how well this image represents the rest of the set (see Algorithm 1). In this framework, only score$(\mathbf{x}_i, \mathcal{D}_{wi})$ varies across the different models. We present the different ways to compute this score below.

**Bayesian model.** Since we have already shown the relationship between our rational measure and Bayesian Sets, the score in this model is computed efficiently via Equation 2. The hyperparameters $\alpha$ and $\beta$ are set empirically from the entire dataset, $\alpha = \kappa \mathbf{m}$, $\beta = \kappa(\mathbf{1} - \mathbf{m})$, where $\mathbf{m}$ is the mean of $\mathbf{x}$ over all images, and $\kappa$ is a scaling factor. An example of using this measure on the set of 299 images for category label *coast* is presented in Figure 1. Panels (a) and (b) of this figure show the top nine and bottom nine ranked images, respectively, where it is quite apparent that the top ranked images depict a better set of *coast* examples than the bottom rankings. It also becomes clear how poorly this label applies to some of the images in the bottom rankings, which is an important issue if using the labels provided with the Corel database as part of a training set for learning algorithms.

**Likelihood model.** This model treats representative judgments of an item $\mathbf{x}^*$ as $p(\mathbf{x}^*|\mathcal{D}_s)$ for a set $\mathcal{D}_s = \{\mathbf{x}_1, \ldots, \mathbf{x}_N\}$. Since this probability can also be expressed as $\frac{p(\mathbf{x}^*, \mathcal{D}_s)}{p(\mathcal{D}_s)}$, we can derive an efficient scheme for computing the score similar to the Bayesian Sets scoring criterion by making the same model assumptions. The likelihood model scoring criterion is

$$\text{Lscore}(\mathbf{x}^*) = \frac{p(\mathbf{x}^*, \mathcal{D}_s)}{p(\mathcal{D}_s)} = \prod_j \frac{1}{\alpha_j + \beta_j + N} \left(\tilde{\alpha}_j\right)^{x_{*j}} \left(\tilde{\beta}_j\right)^{1 - x_{*j}} \tag{6}$$

where $\tilde{\alpha}_j = \alpha_j + \sum_{n=1}^{N} x_{nj}$ and $\tilde{\beta}_j = \beta_j + N - \sum_{n=1}^{N} x_{nj}$. The logarithm of this score is also linear in $\mathbf{x}$ and can be computed efficiently as

$$\log \text{Lscore}(\mathbf{x}^*) = c + \sum_j w_j x_{*j} \tag{7}$$

where $c = \sum_j \log \beta_j - \log(\alpha_j + \beta_j + N)$ and $w_j = \log \tilde{\alpha}_j - \log \tilde{\beta}_j$. The hyperparameters $\alpha$ and $\beta$ are initialized to the same values used in the Bayesian model.

**Prototype model.** In this model we define a prototype vector $\mathbf{x}_{\text{proto}}$ to be the modal features for a set of items $\mathcal{D}_s$. The similarity measure then becomes

$$\text{Pscore}(\mathbf{x}^*) = \exp\{-\lambda \operatorname{dist}(\mathbf{x}^*, \mathbf{x}_{\text{proto}})\} \tag{8}$$

where $\operatorname{dist}(\cdot, \cdot)$ is the Hamming distance between the two vectors and $\lambda$ is a free parameter. Since we are primarily concerned with ranking images, $\lambda$ does not need to be optimized as it plays the role of a scaling constant.

**Exemplar model.** We define the exemplar model using a similar scoring metric to the prototype model, except rather than computing the distance of $\mathbf{x}^*$ to a single prototype, we compute a distance for each item in the set $\mathcal{D}_s$. Our similarity measure is thus computed as

$$\text{Escore}(\mathbf{x}^*) = \sum_{x_j \in \mathcal{D}_s} \exp\{-\lambda \operatorname{dist}(\mathbf{x}^*, \mathbf{x}_j)\} \tag{9}$$

where $\operatorname{dist}(\cdot, \cdot)$ is the Hamming distance between two vectors and $\lambda$ is a free parameter. In this case, $\lambda$ does need to be optimized as the sum means that different values for $\lambda$ can result in different overall similarity scores.

## 5 Modeling Human Ratings of Representativeness

Given a set of images provided with a category label, how do people determine which images are good or bad examples of that category? In this section we present an experiment which evaluates our models through comparison with human judgments of the representativeness of images.

### 5.1 Methods

A total of 500 participants (10 per category) were recruited via Amazon Mechanical Turk and compensated \$0.25. The stimuli were created by identifying the top 10 and bottom 10 ranked images for each of the 50 categories for the Bayesian, likelihood, and prototype models and then taking the union of these sets for each category. The exemplar model was excluded in this process as it required optimization of its $\lambda$ parameter, meaning that the best and worst images could not be determined in advance. The result was a set of 1809 images, corresponding to an average of 36 images per category. Participants were shown a series of images and asked to rate how good an example each image was of the assigned category label. The order of images presented was randomized across subjects. Image quality ratings were made on a scale of 1-7, with a rating of 1 meaning the image is a very bad example and a rating of 7 meaning the image is a very good example.

### 5.2 Results

Once the human ratings were collected, we computed the mean ratings for each image and the mean of the top 10 and bottom 10 results for each algorithm used to create the stimuli. We also computed

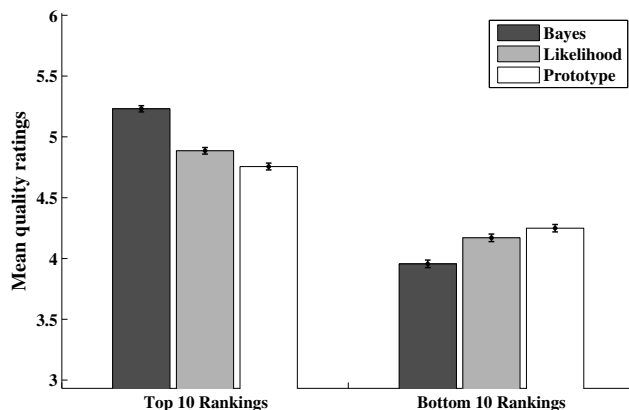

Figure 2: Mean quality ratings of the top 10 and bottom 10 rankings of the different representativeness models over 50 categories. Error bars show one standard error. The vertical axis is bounded by the best possible top 10 ratings and the worst possible bottom 10 ratings across categories.

bounds for the ratings based on the optimal set of top 10 and bottom 10 images per category. These are the images which participants rated highest and lowest, regardless of which algorithm was used to create the stimuli. The mean ratings for the optimal top 10 images was slightly less than the highest possible rating allowed ($m = 6.018$, $se = 0.074$), while the mean ratings for the optimal bottom 10 images was significantly higher than the lowest possible rating allowed ($m = 2.933$, $se = 0.151$). The results are presented in Figure 2. The Bayesian model had the overall highest ratings for its top 10 rankings ($m = 5.231$, $se = 0.026$) and the overall lowest ratings for its bottom 10 rankings ($m = 3.956$, $se = 0.031$). The other models performed significantly worse, with likelihood giving the next highest top 10 ($m = 4.886, se = 0.028$), and next lowest bottom 10 ($m = 4.170, se = 0.031$), and prototype having the lowest top 10 ($m = 4.756$, $se = 0.028$), and highest bottom 10 ($m = 4.249$, $se = 0.031$). We tested for statistical significance via pairwise t-tests on the mean differences of the top and bottom 10 ratings over all 50 categories, for each pair of models. The Bayesian model outperformed both other algorithms ($p < .001$).

As a second analysis, we ran a Spearman rank-order correlation to examine how well the actual scores from the models fit with the entire set of human judgments. Although we did not explicitly ask participants to rank images, their quality ratings implicitly provide an ordering on the images that can be compared against the models. This also gives us an opportunity to evaluate the exemplar model, optimizing its $\lambda$ parameter to maximize the fit to the human data. To perform this correlation we recorded the model scores over all images for each category, and then computed the correlation of each model with the human judgments within that category. Correlations were then averaged across categories. The Bayesian model had the best mean correlation ($\rho = 0.352$), while likelihood ($\rho = 0.220$), prototype ($\rho = 0.160$), and the best exemplar model ($\lambda = 2.0$, $\rho = 0.212$) all performed less well. Paired t-tests showed that the Bayesian model produced statistically significantly better performance than the other three models (all $p < .01$).

## 5.3 Discussion

Overall, the Bayesian model of representativeness provided the best account of people's judgments of which images were good and bad examples of the different categories. The mean ratings over the entire dataset were best predicted by our model, indicating that on average, the model predictions for images in the top 10 results were deemed of high quality and the predictions for images in the bottom 10 results were deemed of low quality. Since the images from the Corel database come with labels given by human judges, few images are actually very bad examples of their prescribed labels. This explains why the ratings for the bottom 10 images are not much lower. Additionally, there was some variance as to which images the Mechanical Turk workers considered to be "most representative". This explains why the ratings for the top 10 images are not much higher, and thus why the difference between top and bottom 10 on average is not larger. When comparing the actual

Table 1: Model comparisons for the outlier experiment

| Model | Average Outlier Position | S.E. |
|---|---|---|
| Bayesian Sets | 0.805 | ± 0.014 |
| Likelihood | 0.779 | ± 0.013 |
| Prototype | 0.734 | ± 0.015 |
| Exemplar | 0.734 | ± 0.016 |

scores from the different models against the ranked order of human quality ratings, the Bayesian account was also significantly more accurate than the other models. While the actual correlation value was less than 1, the dataset was rather varied in terms of quality for each category and thus it was not expected to be a perfect correlation. The methods of the experiment were also not explicitly testing for this effect, providing another source of variation in the results.

## 6 Finding Outliers in Sets

Measuring the representativeness of items in sets can also provide a novel method of finding outliers in sets. Outliers are defined as an observation that appears to deviate markedly from other members of the sample in which it occurs [17]. Since models of representativeness can be used to rank items in a set by how good an example they are of the entire set, outliers should receive low rankings. The performance of these different measures in detecting outliers provides another indirect means of assessing their quality as measures of representativeness.

To empirically test this idea we can take an image from a particular category and inject it into all other categories, and see whether the different measures can identify it as an outlier. To find a good candidate image we used the top ranking image per category as ranked by the Bayesian model. We justify this method because the Bayesian model had the best performance in predicting human quality judgments. Thus, the top ranked image for a particular category is assumed to be a bad example of the other categories. We evaluated how low this outlier was ranked by each of the representativeness measures 50 times, testing the models with a single injected outlier from each category to get a more robust measure. The final evaluation was based on the normalized outlier ranking for each category (position of outlier divided by total number of images in the category), averaged over the 50 injections. The closer this quantity is to 1, the lower the ranking of outliers.

The results of this analysis are depicted in Table 1, where it can be seen that the Bayesian model outperforms the other models. It is interesting to note that these measures are all quite distant from 1. We interpret this as another indication of the noisiness of the original image labels in the dataset since there were a number of images in each category that were ranked lower than the outlier.

## 7 Conclusions

We have extended an existing Bayesian model of representativeness to handle sets of items and showed how it closely approximates a method of clustering on-demand – Bayesian Sets – that had been developed in machine learning. We exploited this relationship to allow us to evaluate a set of psychological models of representativeness using a large database of naturalistic images. Our Bayesian measure of representativeness significantly outperformed other proposed accounts in predicting human judgments of how representative images were of different categories. These results provide strong evidence for this characterization of representativeness, and a new source of validation for the Bayesian Sets algorithm. We also introduced a novel method of detecting outliers in sets of data using our representativeness measure, and showed that it outperformed other measures. We hope that the combination of methods from cognitive science and computer science that we used to obtain these results is the first step towards closer integration between these disciplines, linking psychological theories and behavioral methods to sophisticated algorithms and large databases.

**Acknowledgments.** This work was supported by grants IIS-0845410 from the National Science Foundation and FA-9550-10-1-0232 from the Air Force Office of Scientific Research to TLG and a National Science Foundation Postdoctoctoral Fellowship to KAH.

# References

[1] D. Kahneman and A. Tversky. Subjective probability: A judgment of representativeness. *Cognitive Psychology*, 3:430–454, 1972.

[2] G. Gigerenzer. On narrow norms and vague heuristics: A reply to Kahneman and Tversky (1996). *Psychological Review*, 103:592, 1996.

[3] G. L. Murphy and D. L. Medin. The role of theories in conceptual coherence. *Psychological Review*, 92:289–316, 1985.

[4] J. B. Tenenbaum and T. L. Griffiths. The rational basis of representativeness. In *Proc. 23rd Annu. Conf. Cogn. Sci. Soc.*, pages 1036–1041, 2001.

[5] Z. Ghahramani and K. A. Heller. Bayesian sets. In *Advances in Neural Information Processing Systems*, volume 18, 2005.

[6] C.B. Mervis and E. Rosch. Categorization of natural objects. *Annual Review of Psychology*, 32:89–115, 1981.

[7] D.N. Osherson, E.E. Smith, O. Wilkie, A. Lopez, and E. Shafir. Category-based induction. *Psychological Review*, 97:185, 1990.

[8] D. L. Medin and M. M. Schaffer. Context theory of classification learning. *Psychological Review*, 85:207–238, 1978.

[9] R. M. Nosofsky. Attention and learning processes in the identification and categorization of integral stimuli. *Journal of Experimental Psychology: Learning, Memory, and Cognition*, 13:87–108, 1987.

[10] S. K. Reed. Pattern recognition and categorization. *Cognitive Psychology*, 3:393–407, 1972.

[11] G. Gigerenzer and U. Hoffrage. How to improve Bayesian reasoning without instruction: Frequency formats. *Psychological Review*, 102:684, 1995.

[12] R. N. Shepard. Towards a universal law of generalization for psychological science. *Science*, 237:1317–1323, 1987.

[13] J. R. Anderson. *The adaptive character of thought*. Erlbaum, Hillsdale, NJ, 1990.

[14] K. A. Heller and Z. Ghahramani. A simple Bayesian framework for content-based image retrieval. *IEEE Conference on Computer Vision and Pattern Recognition*, 2:2110–2117, 2006.

[15] R. Silva, K. A. Heller, and Z. Ghahramani. Analogical reasoning with relational Bayesian sets. *International Conference on AI and Statistics*, 2007.

[16] H. Müller, S. Marchand-Maillet, and T. Pun. The truth about Corel - evaluation in image retrieval. *International Conference on Image and Video Retrieval*, 2002.

[17] F. Grubbs. Procedures for detecting outlying observations in samples. *Technometrics*, 11:1–21, 1969.

